# Hidden Markov Dirichlet Process: Modeling Genetic Recombination in Open Ancestral Space

**Kyung-Ah Sohn**
School of Computer Science
Carnegie Mellon University
Pittsburgh, PA 15213
ksohn@cs.cmu.edu

**Eric P. Xing**
School of Computer Science
Carnegie Mellon University
Pittsburgh, PA 15213
epxing@cs.cmu.edu

## Abstract

We present a new statistical framework called hidden Markov Dirichlet process (HMDP) to jointly model the genetic recombinations among possibly infinite number of founders and the coalescence-with-mutation events in the resulting genealogies. The HMDP posits that a haplotype of genetic markers is generated by a sequence of recombination events that select an ancestor for each locus from an unbounded set of founders according to a 1st-order Markov transition process. Conjoining this process with a mutation model, our method accommodates both between-lineage recombination and within-lineage sequence variations, and leads to a compact and natural interpretation of the population structure and inheritance process underlying haplotype data. We have developed an efficient sampling algorithm for HMDP based on a two-level nested Pólya urn scheme. On both simulated and real SNP haplotype data, our method performs competitively or significantly better than extant methods in uncovering the recombination hotspots along chromosomal loci; and in addition it also infers the ancestral genetic patterns and offers a highly accurate map of ancestral compositions of modern populations.

## 1 Introduction

Recombinations between ancestral chromosomes during meiosis play a key role in shaping the patterns of linkage disequilibrium (LD)—the non-random association of alleles at different loci—in a population. When a recombination occurs between two loci, it tends to decouple the alleles carried at those loci in its descendants and thus reduce LD; uneven occurrence of recombination events along chromosomal regions during genetic history can lead to "block structures" in molecular genetic polymorphisms such that within each block only low level of diversities are present in a population. The problem of inferring chromosomal recombination hotspots is essential for understanding the origin and characteristics of genome variations; several combinatorial and statistical approaches have been developed for uncovering optimum block boundaries from single nucleotide polymorphism (SNP) haplotypes [Daly *et al.*, 2001; Anderson and Novembre, 2003; Patil *et al.*, 2001; Zhang *et al.*, 2002], and these advances have important applications in genetic analysis of disease propensities and other complex traits. The deluge of SNP data also fuels the long-standing interest of analyzing patterns of genetic variations to reconstruct the evolutionary history and ancestral structures of human populations, using, for example, variants of admixture models on genetic polymorphisms [Rosenberg *et al.*, 2002]. These progress notwithstanding, the statistical methodologies developed so far mostly deal with LD analysis and ancestral inference separately, using specialized models that do not capture the close statistical and genetic relationships of these two problems. Moreover, most of these approaches ignore the inherent uncertainty in the genetic complexity (e,g., the number of genetic founders of a population) of the data and rely on inflexible models built on a pre-fixed, closed genetic space. Recently, Xing *et al.* [2004; 2006] have developed a nonparametric Bayesian framework for modeling genetic polymorphisms based on the Dirichlet process mixtures and extensions, which attempts to allow more flexible control over the number of genetic founders than has been provided by the statistical methods proposed thus far. In this paper, we leverage on this approach and present a unified framework to model complex genetic inheritance process that allows recombinations among possibly infinite founding alleles and coalescence-with-mutation events in the resulting genealogies.

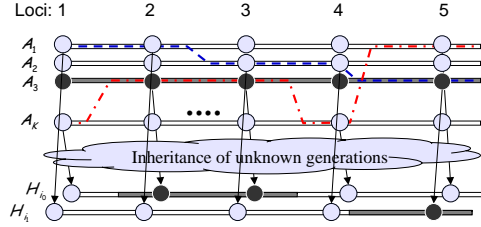

Figure 1: An illustration of a hidden Markov Dirichlet process for haplotype recombination and inheritance.

We assume that individual chromosomes in a modern population are originated from an unknown number of ancestral haplotypes via biased random recombinations and mutations (Fig 1). The recombinations between the ancestors follow a a state-transition process we refer to as hidden Markov Dirichlet process (originated from the infinite HMM by Beal *et al.* [2001]), which travels in an open ancestor space, with nonstationary recombination rates depending on the genetic distances between SNP loci. Our model draws inspiration from the HMM proposed in [Greenspan and Geiger, 2003], but we employ a two-level Pólya urn scheme akin to the hierarchical DP [Teh *et al.*, 2004] to accommodate an open ancestor space, and allow full posterior inference of the recombination sites, mutation rates, haplotype origin, ancestor patterns, etc., conditioning on phased SNP data, rather than estimating them using information theoretic or maximum likelihood principles. On both simulated and real genetic data, our model and algorithm show competitive or superior performance on a number of genetic inference tasks over the state-of-the-art parametric methods.

## 2 Hidden Markov Dirichlet Process for Recombination

Sequentially choosing recombination targets from a set of ancestral chromosomes can be modeled as a hidden Markov process [Niu *et al.*, 2002; Greenspan and Geiger, 2003], in which the hidden states correspond to the index of the candidate chromosomes, the transition probabilities correspond to the recombination rates between the recombining chromosome pairs, and the emission model corresponds to a mutation process that passes the chosen chromosome region in the ancestors to the descents. When the number of ancestral chromosomes is not known, it is natural to consider an HMM whose state space is countably infinite [Beal *et al.*, 2001; Teh *et al.*, 2004]. In this section, we describe such an infinite HMM formalism, which we would like to call *hidden Markov Dirichlet process*, for modeling recombination in an open ancestral space.

### 2.1 Dirichlet Process mixtures

For self-containedness, we begin with a brief recap of the basic Dirichlet process mixture model proposed in Xing *et al.* [2004] for haplotype inheritance without recombination. A *haplotype* refers to the joint allele configuration of a contiguous list of SNPs located on a single chromosome (Fig 1). Under a well-known genetic model known as *coalescence-with-mutation* (but without recombination), one can treat a haplotype from a modern individual as a descendent of an unknown ancestor haplotype (i.e., a founder) via random mutations that alter the allelic states of some SNPs. It can be shown that such a coalescent process in an infinite population leads to a partition of the population that can be succinctly captured by the following Pólya urn scheme. Consider an urn that at the outset contains a ball of a single color. At each step we either draw a ball from the urn and replace it with two balls of the same color, or we are given a ball of a new color which we place in the urn. One can see that such a scheme leads to a partition of the balls according to their color. Letting parameter $\tau$ define the probabilities of the two types of draws, and viewing each (distinct) color as a sample from $Q_0$, and each ball as a sample from $Q$, Blackwell and MacQueen [1973] showed that this Pólya urn model yields samples whose distributions are those of the marginal probabilities under the *Dirichlet process*. One can associate mixture component with colors in the Pólya urn model, and thereby define a "clustering" of the data. The resulting model is known as a *DP mixture*. Note that a DP mixture requires no prior specification of the number of components. Back to haplotype modeling, following Xing *et al.* [2004; 2006], let $H_i = [H_{i,1}, \ldots, H_{i,T}]$ denote a haplotype over $T$ SNPs from chromosome $i$ [1]; let $A_k = [A_{k,1}, \ldots, A_{k,T}]$ denote an ancestor haplotype (indexed by $k$) and $\theta_k$ denote the *mutation rate* of ancestor $k$; and let $C_i$ denote an *inheritance variable* that specifies the ancestor of haplotype $H_i$. As described in Xing *et al.* [2006], under a DP mixture, we have the following Pólya urn

scheme for sampling modern haplotypes:

- Draw first haplotype:

$a_1 \mid \mathrm{DP}(\tau, Q_0) \sim Q_0(\cdot),$      sample the 1st founder;

$h_1 \sim P_h(\cdot \mid a_1, \theta_1),$      sample the 1st haplotype from an inheritance model defined on the 1st founder;

- for subsequent haplotypes:
  - sample the founder indicator for the $i$th haplotype:

$$c_i \mid \mathrm{DP}(\tau, Q_0) \sim \begin{cases} p(c_i = c_j \text{ for some } j < i \mid c_1, \ldots, c_{i-1}) = \frac{n_{c_j}}{i-1+\tau} \\ p(c_i \neq c_j \text{ for all } j < i \mid c_1, \ldots, c_{i-1}) = \frac{\tau}{i-1+\tau} \end{cases}$$

where $n_{c_i}$ is the *occupancy number* of class $c_i$—the number of previous samples belonging to class $c_i$.

  - sample the founder of haplotype $i$ (indexed by $c_i$):

$$\phi_{c_i} \mid \mathrm{DP}(\tau, Q_0) \begin{cases} = \{a_{c_j}, \theta_{c_j}\} \text{ if } c_i = c_j \text{ for some } j < i \text{ (i.e., } c_i \text{ refers to an inherited founder)} \\ \sim Q_0(a, \theta) \text{ if } c_i \neq c_j \text{ for all } j < i \text{ (i.e., } c_i \text{ refers to a new founder)} \end{cases}$$

  - sample the haplotype according to its founder:

$h_i \mid c_i \sim P_h(\cdot \mid a_{c_i}, \theta_{c_i}).$

Notice that the above generative process assumes each modern haplotype to be originated from a single ancestor, this is only plausible for haplotypes spanning a short region on a chromosome. Now we consider long haplotypes possibly bearing multiple ancestors due to recombinations between an unknown number of founders.

## 2.2 Hidden Markov Dirichlet Process (HMDP)

In a standard HMM, state-transitions across a discrete time- or space-interval take place in a fixed-dimensional state space, thus it can be fully parameterized by, say, a $K$-dimensional initial-state probability vector and a $K \times K$ state-transition probability matrix. As first proposed in Beal *et al.* [2001], and later discussed in Teh *et al.* [2004], one can "open" the state space of an HMM by treating the now infinite number of discrete states of the HMM as the support of a DP, and the transition probabilities to these states from some source as the masses associated with these states. In particular, for each source state, the possible transitions to the target states need to be modeled by a unique DP. Since all possible source states and target states are taken from the same infinite state space, overall we need an open set of DPs with different mass distributions on the SAME support (to capture the fact that different source states can have different transition probabilities to any target state). In the sequel, we describe such a nonparametric Bayesian HMM using an intuitive hierarchical Pólya urn construction. We call this model a **hidden Markov Dirichlet process**.

In an HMDP, both the columns and rows of the transition matrix are infinite dimensional. To construct such an stochastic matrix, we will exploit the fact that in practice only a finite number of states (although we don't know what they are) will be visited by each source state, and we only need to keep track of these states. The following sampling scheme based on a hierarchical Pólya urn scheme captures this spirit and yields a constructive definition of HMDP.

We set up a single "stock" urn at the top level, which contains balls of colors that are represented by at least one ball in one or multiple urns at the bottom level. At the bottom level, we have a set of *distinct* urns which are used to define the initial and transition probabilities of the HMDP model (and are therefore referred as HMM-urns). Specifically, one of HMM urns, $u_0$, is set aside to hold colored balls to be drawn at the onset of the HMM state-transition sequence. Each of the remaining HMM urns is painted with a color represented by at least one ball in the stock urn, and is used to hold balls to be drawn during the execution of a Markov chain of state-transitions. Now let's suppose that at time $t$ the stock urn contains $n$ balls of $K$ distinct colors indexed by an integer set $\mathcal{C} = \{1, 2, \ldots, K\}$; the number of balls of color $k$ in this urn is denoted by $n_k, k \in \mathcal{C}$. For urn $u_0$ and urns $u_1, \ldots, u_K$, let $m_{j,k}$ denote the number of balls of color $k$ in urn $u_j$, and $m_j = \sum_{k \in \mathcal{C}} m_{j,k}$ denote the total number of balls in urn $u_j$. Suppose that at time $t-1$, we had drawn a ball with color $k'$. Then at time $t$, we either draw a ball randomly from urn $u_{k'}$, and place back two balls both of that color; or with probability $\frac{\tau}{m_j + \tau}$ we turn to the top level. From the stock urn, we can either draw a ball randomly and put back two balls of that color to the stock urn and one to $u_{k'}$, or obtain a ball of a new color $K+1$ with probability $\frac{\gamma}{n+\gamma}$ and put back a ball of this color to both the stock urn and urn $u_{k'}$ of the lower level. Essentially, we have a master DP (the stock urn) that serves as a base measure for infinite number of child DPs (HMM-urns). As pointed out in Teh *et al.* [2004], this model can be viewed as an instance of the hierarchical Dirichlet process mixture model.

As discussed in Xing *et al.* [2006], associating each color $k$ with an ancestor configuration $\phi_k = \{a_k, \theta_k\}$ whose values are drawn from the base measure $F \equiv Beta(\theta)p(a)$, conditioning on the Dirichlet process underlying the stock urn, the samples in the $j$th bottom-level urn are also distributed as marginals under a Dirichlet measure:

$$\phi_{m_j}|\phi_{-m_j} \sim \sum_{k=1}^{K} \frac{m_{j,k} + \tau\frac{n_k}{n-1+\gamma}}{m_j - 1 + \tau} \delta_{\phi_k^*}(\phi_{m_j}) + \frac{\tau}{m_j - 1 + \tau}\frac{\gamma}{n-1+\gamma}F(\phi_{m_j})$$

$$= \sum_{k=1}^{K} \pi_{j,k}\delta_{\phi_k^*}(\phi_{m_j}) + \pi_{j,K+1}F(\phi_{m_j}), \tag{1}$$

where $\pi_{j,k} \equiv \frac{m_{j,k}+\tau\frac{n_k}{n-1+\gamma}}{m_j-1+\tau}$, $\pi_{j,K+1} \equiv \frac{\tau}{m_j-1+\tau}\frac{\gamma}{n-1+\gamma}$. Let $\pi_j \equiv [\pi_{j,1},\pi_{j,2},\ldots]$, now we have an infinite-dimensional Bayesian HMM that, given $F,\gamma,\tau$, and all initial states and transitions sampled so far, follows an initial states distribution parameterized by $\pi_0$, and transition matrix $\Pi$ whose rows are defined by $\{\pi_j : j > 0\}$. As in Xing *et al.* [2006], we also introduce vague inverse Gamma priors for the concentration parameters $\gamma$ and $\tau$.

### 2.3 HMDP Model for Recombination and Inheritance

Now we describe a stochastic model, based on an HMDP, for generating individual haplotypes in a modern population from a hypothetical pool of ancestral haplotypes via recombination and mutations (i.e., random mating with neutral selection). For each modern chromosome $i$, let $C_i = [C_{i,1},\ldots,C_{i,T}]$ denote the sequence of inheritance variables specifying the index of the ancestral chromosome at each SNP locus. When no recombination takes place during the inheritance process that produces haplotype $H_i$ (say, from ancestor $k$), then $C_{i,t} = k, \forall t$. When a recombination occurs, say, between loci $t$ and $t+1$, we have $C_{i,t} \neq C_{i,t+1}$. We can introduce a Poisson point process to control the duration of non-recombinant inheritance. That is, given that $C_{i,t} = k$, then with probability $e^{-dr}+(1-e^{-dr})\pi_{kk}$, where $d$ is the physical distance between two loci, $r$ reflects the rate of recombination per unit distance, and $\pi_{kk}$ is the self-transition probability of ancestor $k$ defined by HMDM, we have $C_{i,t+1} = C_{i,t}$; otherwise, the source state (i.e., ancestor chromosome $k$) pairs with a target state (e.g., ancestor chromosome $k'$) between loci $t$ and $t+1$, with probability $(1-e^{-dr})\pi_{kk'}$. Hence, each haplotype $H_i$ is a mosaic of segments of multiple ancestral chromosomes from the ancestral pool $\{A_{k,\cdot}\}_{k=1}^{\infty}$. Essentially, the model we described so far is a time-inhomogeneous infinite HMM. When the physical distance information between loci is not available, we can simply set $r$ to be infinity so that we are back to a standard stationary HMDP model.

The emission process of the HMDM corresponds to an inheritance model from an ancestor to the matching descendent. For simplicity, we adopt the *single-locus mutation model* in Xing *et al.* [2004]:

$$p(h_t|a_t,\theta) = \theta^{\mathbb{I}(h_t=a_t)}\left(\frac{1-\theta}{|B|-1}\right)^{\mathbb{I}(h_t\neq a_t)}, \tag{2}$$

where $h_t$ and $a_t$ denote the alleles at locus $t$ of an individual haplotype and its corresponding ancestor, respectively; $\theta$ indicates the ancestor-specific mutation rate; and $|B|$ denotes the number of possible alleles. As discussed in Liu *et al.* [2001], this model corresponds to a star genealogy resulted from infrequent mutations over a shared ancestor, and is widely used in statistical genetics as an approximation to a full coalescent genealogy. Following Xing *et al.* [2004], assume that the mutation rate $\theta$ admits a Beta prior, the marginal conditional likelihood of a haplotype given its matching ancestor can be computed by integrating out $\theta$ under the Bayesian rule.

## 3 Posterior Inference

Now we proceed to describe a Gibbs sampling algorithm for posterior inference under HMDP. The variables of interest include $\{C_{i,t}\}$, the inheritance variables specifying the origins of SNP alleles of all loci on each haplotype; and $\{A_{k,t}\}$, the founding alleles at all loci of each ancestral haplotype.

The Gibbs sampler alternates between two sampling stages. First it samples the inheritance variables $\{c_{i,t}\}$, conditioning on all given individual haplotypes $\mathbf{h} = \{h_1,\ldots,h_{2N}\}$, and the most recently sampled configuration of the ancestor pool $\mathbf{a} = \{a_1,\ldots,a_K\}$; then given $\mathbf{h}$ and current values of the $c_{i,t}$'s, it samples every ancestor $a_k$.

To improve the mixing rate, we sample the inheritance variables one block at a time. That is, every time we sample $\delta$ consecutive states $c_{t+1},\ldots,c_{t+\delta}$ starting at a randomly chosen locus $t+1$ along a haplotype. (For simplicity we omit the haplotype index $i$ here and in the forthcoming expositions when it is clear from context that the statements or formulas apply to all individual haplotypes.) Let $\mathbf{c}^-$ denote the set of previously sampled inheritance variables. Let $\mathbf{n}$ denote the totality of occupancy records of the top-level DP (i.e. the "stock urn") — $\{n\} \cup \{n_k : \forall k\}$; and $\mathbf{m}$ denote the totality of the occupancy records of each lower-level DPs (i.e., the urns corresponding to the recombination choices by each ancestor) — $\{m_k : \forall k\} \cup \{m_{k,k'} : \forall k,k'\}$. And let $\mathbf{l}_k$ denote the sufficient statistics associated with all haplotype instances originated from ancestor $k$. The predictive

distribution of a $\delta$-block of inheritance variables can be written as:

$$p(c_{t+1:t+\delta} \,|\mathbf{c}^-, \mathbf{h}, \mathbf{a}) \quad \propto \quad p(c_{t+1:t+\delta} \,|c_t, c_{t+\delta+1}, \mathbf{m}, \mathbf{n})p(h_{t+1:t+\delta}|a_{c_{t+1},t+1}, \ldots, a_{c_{t+\delta},t+\delta})$$

$$\propto \quad \prod_{j=t}^{t+\delta} p(c_{j+1}|c_j, \mathbf{m}, \mathbf{n}) \prod_{j=t+1}^{t+\delta} p(h_j|a_{c_j,j}, \mathbf{l}_{c_j}). \tag{3}$$

This expression is simply Bayes' theorem with $p(h_{t+1:t+\delta}|a_{c_{t+1},t+1}, \ldots, a_{c_{t+\delta},t+\delta})$ playing the role of the likelihood and $p(c_{t+1:t+\delta} \,|\mathbf{c}^-, \mathbf{h}, \mathbf{a})$ playing the role of the prior. One should be careful that the sufficient statistics $\mathbf{n}$, $\mathbf{m}$ and $\mathbf{l}$ employed here should exclude the contributions by samples associated with the $\delta$-block to be sampled. Note that naively, the sampling space of an inheritance block of length $\delta$ is $|A|^\delta$ where $|A|$ represents the cardinality of the ancestor pool. However, if we assume that the recombination rate is low and block length is not too big, then the probability of having two or more recombination events within a $\delta$-block is very small and thus can be ignored. This approximation reduces the sampling space of the $\delta$-block to $O(|A|\delta)$, i.e., $|A|$ possible recombination targets times $\delta$ possible recombination locations. Accordingly, Eq. (3) reduces to:

$$p(c_{t+1:t+\delta} \,|\mathbf{c}^-, \mathbf{h}, \mathbf{a}) \propto p(c_{t'} \,|c_{t'-1} = c_t, \mathbf{m}, \mathbf{n})p(c_{t+\delta+1} \,|c_{t+\delta} = c_{t'}, \mathbf{m}, \mathbf{n}) \prod_{j=t'}^{t+\delta} p(h_j|a_{c_{t'},j}, \mathbf{l}_{c_{t'}})$$

for some $t' \in [t+1, t+\delta]$. Recall that in an HMDP model for recombination, given that the total recombination probability between two loci $d$-units apart is $\lambda \equiv 1 - e^{-dr} \approx dr$ (assuming $d$ and $r$ are both very small), the transition probability from state $k$ to state $k'$ is:

$$p(c_{t'} = k' \,|c_{t'-1} = k, \mathbf{m}, \mathbf{n}, r, d)$$
$$= \begin{cases} \lambda\pi_{k,k'} + (1-\lambda)\delta(k,k') & \text{for } k' \in \{1, ..., K\}, \text{ i.e., transition to an existing ancestor,} \\ \lambda\pi_{k,K+1} & \text{for } k' = K+1, \text{ i.e., transition to a new ancestor,} \end{cases} \tag{4}$$

where $\pi_k$ represents the transition probability vector for ancestor $k$ under HMDP, as defined in Eq. (1). Note that when a new ancestor $a_{K+1}$ is instantiated, we need to immediately instantiate a new DP under $F$ to model the transition probabilities from this ancestor to all instantiated ancestors (including itself). Since the occupancy record of this DP, $\mathbf{m}_{K+1} := \{m_{K+1}\} \cup \{m_{K+1,k} : k = 1, \ldots, K+1\}$, is not yet defined at the onset, with probability 1 we turn to the top-level DP when departing from state $K+1$ for the first time. Specifically, we define $p(\cdot|c_{t'} = K+1)$ according to the occupancy record of ancestors in the stock urn. For example, at the distal boarder of the $\delta$-block, since $c_{t+\delta+1}$ always indexes a previously inherited ancestor (and therefore must be present in the stock-urn), we have:

$$p(c_{t+\delta+1} \,|c_{t+\delta} = K+1, \mathbf{m}, \mathbf{n}) \quad = \quad \lambda \times \frac{n_{c_{t+\delta+1}}}{n + \gamma}. \tag{5}$$

Now we can substitute the relevant terms in Eq. (3) with Eqs. (4) and (5). The marginal likelihood term in Eq. (3) can be readily computed based on Eq. (2), by integrating out the mutation rate $\theta$ under a Beta prior (and also the ancestor $a$ under a uniform prior if $c_{t'}$ refers to an ancestor to be newly instantiated) [Xing *et al.*, 2004]. Putting everything together, we have the proposal distribution for a block of inheritance variables. Upon sampling every $c_t$, we update the sufficient statistics $\mathbf{n}$, $\mathbf{m}$ and $\{\mathbf{l}_k\}$ as follows. First, before drawing the sample, we erase the contribution of $c_t$ to these sufficient statistics. In particular, if an ancestor gets no occupancy in either the stock and the HMM urns afterwards, we remove it from our repository. Then, after drawing a new $c_t$, we increment the relevant counts accordingly. In particular, if $c_t = K+1$ (i.e., a new ancestor is to be drawn), we update $n = n+1$, set $n_{K+1} = 1$, $m_{c_t} = m_{c_t} + 1$, $m_{c_t,K+1} = 1$, and set up a new (empty) HMM urn with color $K+1$ (i.e. instantiating $\mathbf{m}_{K+1}$ with all elements equal to zero).

Now we move on to sample the founders $\{a_{k,t}\}$, following the same proposal given in Xing *et al.* [2006], which is adapted below for completeness:

$$p(a_{k,t}|\mathbf{c}, \mathbf{h}) \propto \prod_{i,t|c_{i,t}=k} p(h_{i,t}|a_{k,t}) = \frac{\Gamma(\alpha_h + l_{k,t})\Gamma(\beta_h + l'_{k,t})}{\Gamma(\alpha_h + \beta_h + l_{k,t} + l'_{k,t})(|B|-1)^{l'_{k,t}}}R(\alpha_h, \beta_h), \tag{6}$$

where $l_{k,t}$ is the number of allelic instances originating from ancestor $k$ at locus $t$ that are identical to the ancestor, when the ancestor has the pattern $a_{k,t}$; and $l'_{k,t} = \sum_i \mathbb{I}(c_{i,t} = k|a_{k,t}) - l_{k,t}$ represents the complement. If $k$ is not represented previously, we can just set $l_{k,t}$ and $l'_{k,t}$ both to zero. Note that when sampling a new ancestor, we can only condition on a small segment of an individual haplotype. To instantiate a complete ancestor, after sampling the alleles in the ancestor corresponding to the segment according to Eq. (6), we first fill in the rest of the loci with random alleles. When another

segment of an individual haplotype needs a new ancestor, we do not naively create a new full-length ancestor; rather, we use the *empty* slots (those with random alleles) of one of the previously instantiated ancestors, if any, so that the number of ancestors does not grow unnecessarily.

## 4 Experiments

We applied the HMDP model to both simulated and real haplotype data. Our analyses focus on the following three popular problems in statistical genetics: 1. Ancestral Inference: estimating the number of founders in a population and reconstructing the ancestor haplotypes; 2) LD-block Analysis: inferring the recombination sites in each individual haplotype and uncover population-level recombination hotspots on the chromosome region; 3) Population Structural Analysis: mapping the genetic origins of all loci of each individual haplotype in a population.

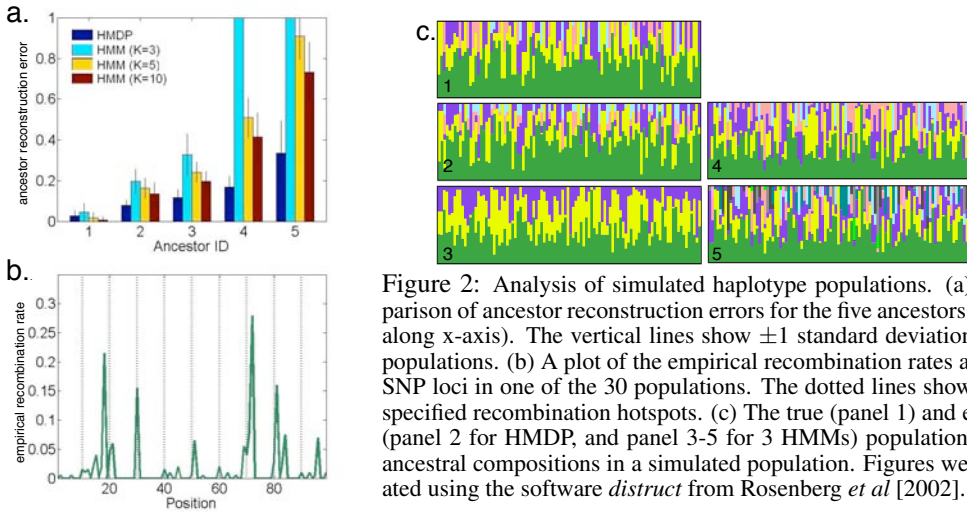

Figure 2: Analysis of simulated haplotype populations. (a) A comparison of ancestor reconstruction errors for the five ancestors (indexed along x-axis). The vertical lines show $\pm 1$ standard deviation over 30 populations. (b) A plot of the empirical recombination rates along 100 SNP loci in one of the 30 populations. The dotted lines show the pre-specified recombination hotspots. (c) The true (panel 1) and estimated (panel 2 for HMDP, and panel 3-5 for 3 HMMs) population maps of ancestral compositions in a simulated population. Figures were generated using the software *distruct* from Rosenberg *et al* [2002].

### 4.1 Analyzing simulated haplotype population

To simulate a population of individual haplotypes, we started with a fixed number, $K_s$ (unknown to the HMDP model), of randomly generated ancestor haplotypes, on each of which a set of recombination hotspots are (randomly) pre-specified. Then we applied a hand-specified recombination process, which is defined by a $K_s$-dimensional HMM, to the ancestor haplotypes to generate $N_s$ individual haplotypes, via sequentially recombining segments of different ancestors according to the simulated HMM states at each locus, and mutating certain ancestor SNP alleles according to the emission model. At the hotspots, we defined the recombination rate to be 0.05, otherwise it is 0.00001. Each individual was forced to have at least one recombination. Overall, 30 datasets each containing 100 individuals (i.e., 200 haplotypes) with 100 SNPs were generated from $K_s = 5$ ancestor haplotypes. As baseline models, we also implemented 3 standard fixed-dimensional HMM, with 3, 5 (the true number of ancestors for the simulated) and 10 hidden states, respectively.

**Ancestral Inference** Using HMDP, we successfully recovered the correct number (i.e., $K = 5$) of ancestors in 21 out of 30 simulated populations; for the remaining 9 populations, we inferred 6 ancestors. From samples of ancestor states $\{a_{k,t}\}$, we reconstructed the ancestral haplotypes under the HMDP model. For comparison, we also inferred the ancestors under the 3 standard HMM using an EM algorithm. We define the *ancestor reconstruction error* $\epsilon_a$ for each ancestor to be the ratio of incorrectly recovered loci over all the chromosomal sites. The average $\epsilon_a$ over 30 simulated populations under 4 different models are shown in Fig 2a. In particular, the average reconstruction errors of HMDP for each of the five ancestors are 0.026, 0.078, 0.116, 0.168, and 0.335, respectively. There is a good correlation between the reconstruction quality and the population frequency of each ancestor. Specifically, the average (over all simulated populations) fraction of SNP loci originated from each ancestor among all loci in the population is 0.472, 0.258, 0.167, 0.068 and 0.034, respectively. As one would expect, the higher the population frequency an ancestor is, the better its reconstruction accuracy. Interestingly, under the fixed-dimensional HMM, even when we use the correct number of ancestor states, i.e., $K = 5$, the reconstruction error is still very high (Fig 2), typically 2.5 times or higher than the error of HMDP. We conjecture that this is because the non-parametric Bayesian treatment of the transition rates and ancestor configurations under the HMDP model leads to a desirable adaptive smoothing effect and also less constraints on the model parameters, which allow them to be more accurately estimated. Whereas under a parametric setting, parameter estimation can easily go sub-optimum due to lack of appropriate smoothing or prior constraints, or deficiency of the learning algorithm (e.g., local-optimality of EM).

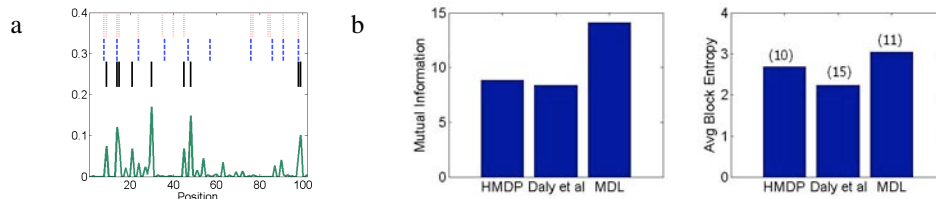

Figure 3: Analysis of the Daly data. (a) A plot of $\lambda_e$ estimated via HMDP; and the haplotype block boundaries according to HMDP (black solid line), HMM [Daly *et al.*, 2001] (red dotted line), and MDL [Anderson and Novembre, 2003]) (blue dashed line). (b) IT scores for haplotype blocks from each method.

**LD-block Analysis** From samples of the inheritance variables $\{c_{i,t}\}$ under HMDP, we can infer the recombination status of each locus of each haplotype. We define the empirical recombination rates $\lambda_e$ at each locus to be the ratio of individuals who had recombinations at that locus over the total number of haploids in the population. Fig 2b shows a plot of the $\lambda_e$ in one of the 30 simulated populations. We can identify the recombination *hotspots* directly from such a plot based on an empirical threshold $\lambda_t$ (i.e., $\lambda_t = 0.05$). For comparison, we also give the true recombination hotspots (depicted as dotted vertical lines) chosen in the ancestors for simulating the recombinant population. The inferred hotspots (i.e., the $\lambda_e$ peaks) show reasonable agreement with the reference.

**Population Structural Analysis** Finally, from samples of the inheritance variables $\{c_{i,t}\}$, we can also uncover the genetic origins of all loci of each individual haplotype in a population. For each individual, we define an empirical *ancestor composition vector* $\eta_e$, which records the fractions of every ancestor in all the $c_{i,t}$'s of that individuals. Fig 2c displays a *population map* constructed from the $\eta_e$'s of all individual. In the population map, each individual is represented by a thin vertical line which is partitioned into colored segments in proportion to the ancestral fraction recorded by $\eta_e$. Five population maps, corresponding to (1) true ancestor compositions, (2) ancestor compositions inferred by HMDP, and (3-5) ancestor compositions inferred by HMMs with 3, 5, 10 states, respectively, are shown in Fig 2c. To assess the accuracy of our estimation, we calculated the distance between the true ancestor compositions and the estimated ones as the mean squared distance between true and the estimated $\eta_e$ over all individuals in a population, and then over all 30 simulated populations. We found that the distance between the HMDP-derived population map and the true map is 0.190, whereas the distance between HMM-map and true map is 0.319, significantly worse than that of HMDP even though the HMM is set to have the true number of ancestral states (i.e., $K = 5$). Because of dimensionality incompatibility and apparent dissimilarity to the true map for other HMMs (i.e., $K = 3$ and 10), we forgo the above quantitative comparison for these two cases.

### 4.2 Analyzing two real haplotype datasets

We applied HMDP to two real haplotype datasets, the single-population Daly data [Daly *et al.*, 2001], and the two-population (CEPH: Utah residents with northern/western European ancestry; and YRI: Yoruba in Ibadan and Nigeria) HapMap data [Thorisson *et al.*, 2005]. These data consist of trios of genotypes, so most of the true haplotypes can be directly inferred from the genotype data.

We first analyzed the 256 individuals from Daly data We compared the recovered recombination hotspots with those reported in Daly *et al.* [2001] (which is based on an HMM employing different number of states at different chromosome segments) and in Anderson and Novembre [2003] (which is based on a minimal description length (MDL) principle). Fig. 3a shows the plot of empirical recombination rates estimated under HMDP, side-by-side with the reported recombination hotspots. There is no ground truth to judge which one is correct; hence we computed information-theoretic (IT) scores based on the estimated within-block haplotype frequencies and the between-block transition probabilities under each model for a comparison. The left panel of Fig 3b shows the total pairwise mutual information between adjacent haplotype blocks segmented by the recombination hotspots uncovered by the three methods. The right panel shows the average entropies of haplotypes within each block. The number above each bar denotes the total number of blocks. The pairwise mutual information score of the HMDP block structure is similar to that of the Daly structure, but smaller than that of MDL. Similar tendencies are observed for average entropies. Note that the Daly and the MDL methods allow the number of haplotype founders to vary across blocks to get the most compact local ancestor constructions. Thus their reported scores might be an underestimate of the true global score because certain segments of an ancestor haplotype that are not or rarely inherited are not counted in the score. Thus the low IT scores achieved by HMDP suggest that HMDP can effectively avoid inferring spurious global and local ancestor patterns. This is confirmed by the population map shown in Fig 4a, which shows that HMDP recovered 6 ancestors and among them the 3 dominant ancestors account for 98% of all the modern haplotypes in the population.

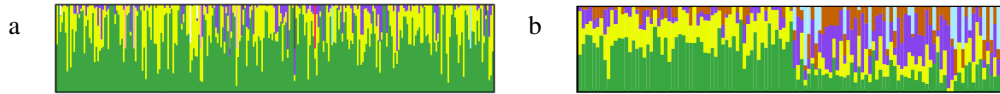
Figure 4: The estimated population maps: (a) Daly data. (b) HapMap data.

The HapMap data contains 60 individuals from CEPH and 60 from YRI. We applied HMDP to the union of the populations, with a random individual order. The two-population structure is clearly retrieved from the population map constructed from the population composition vectors $\eta_e$ for every individual. As seen in Fig. 4b, the left half of the map clearly represents the CEPH population and the right half the YRI population. We found that the two dominant haplotypes covered over 85% of the CEPH population (and the overall breakup among all four ancestors is 0.5618,0.3036,0.0827,0.0518). On the other hand, the frequencies of each ancestor in YRI population are 0.2141,0.1784,0.3209,0.1622,0.1215 and 0.0029, showing that the YRI population is much more diverse than CEPH. Due to space limit, we omit the recombination map of this dataset.

## 5  Conclusion

We have proposed a new Bayesian approach for joint modeling genetic recombinations among possibly infinite founding alleles and coalescence-with-mutation events in the resulting genealogies. By incorporating a hierarchical DP prior to the stochastic matrix underlying an HMM, which facilitates well-defined transition process between infinite ancestor space, our proposed method can efficiently infer a number of important genetic variables, such as recombination hotspot, mutation rates, haplotype origin, and ancestor patterns, jointly under a unified statistical framework.

Emprirically, on both simulated and real data, our approach compares favorably to its parametric counterpart—a fixed-dimensional HMM (even when the number of its hidden state, i.e., the ancestors, is correctly specified) and a few other specialized methods, on ancestral inference, haplotype-block uncovering and population structural analysis. We are interested in further investigating the behavior of an alternative scheme based on reverse-jump MCMC over Bayesian HMMs with different latent states in comparison with HMDP; and we intend to apply our methods to genome-scale LD and demographic analysis using the full HapMap data. While our current model employs only phased haplotype data, it is straightforward to generalize it to unphased genotype data as provided by the HapMap project. HMDP can also be easily adapted to many engineering and information retrieval contexts such as object and theme tracking in open space. Due to space limit, we left out some details of the algorithms and more results of our experiments, which are available in the full version of this paper [Xing and Sohn, 2006].

## Footnotes

[1] We ignore the parental origin index of haplotype as used in Xing *et al.* [2004], and assume that the paternal and maternal haplotypes of each individual are given unambiguously (i.e., *phased*, as known in genetics), as is the case in many LD and haplotype-block analyses. But it is noteworthy that our model can generalize straightforwardly to unphased genotype data by incorporating a simple genotype model as in Xing *et al.* [2004].

## References

[Anderson and Novembre, 2003]  E. C. Anderson and J. Novembre. Finding haplotype block boundaries by using the minimum-description-length principle. *Am J Hum Genet*, 73:336–354, 2003.

[Beal *et al.*, 2001]  M. J. Beal, Z. Ghahramani, and C. E. Rasmussen. The infinite hidden Markov model. In *Advances in Neural Information Processing Systems 13*, 2001.

[Blackwell and MacQueen, 1973]  D. Blackwell and J. B. MacQueen. Ferguson distributions via polya urn schemes. *Annals of Statistics*, 1:353–355, 1973.

[Daly *et al.*, 2001]  M. J. Daly, J. D. Rioux, S. F. Schaffner, T. J. Hudson, and E. S. Lander. High-resolution haplotype structure in the human genome. *Nature Genetics*, 29(2):229–232, 2001.

[Greenspan and Geiger, 2003]  D. Greenspan and D. Geiger. Model-based inference of haplotype block variation. In *Proceedings of RECOMB 2003*, 2003.

[Liu *et al.*, 2001]  J. S. Liu, C. Sabatti, J. Teng, B.J.B. Keats, and N. Risch. Bayesian analysis of haplotypes for linkage disequilibrium mapping. *Genome Res.*, 11:1716–1724, 2001.

[Niu *et al.*, 2002]  T. Niu, S. Qin, X. Xu, and J. Liu. Bayesian haplotype inference for multiple linked single nucleotide polymorphisms. *American Journal of Human Genetics*, 70:157–169, 2002.

[Patil *et al.*, 2001]  N. Patil, A. J. Berno, D. A. Hinds, et al. Blocks of limited haplotype diversity revealed by high-resolution scanning of human chromosome 21. *Science*, 294:1719–1723, 2001.

[Rosenberg *et al.*, 2002]  N. A. Rosenberg, J. K. Pritchard, J. L. Weber, H. M. Cann, K. K. Kidd, L. A. Zhivotovsky, and M. W. Feldman. Genetic structure of human populations. *Science*, 298:2381–2385, 2002.

[Teh *et al.*, 2004]  Y. Teh, M. I. Jordan, M. Beal, and D. Blei. Hierarchical Dirichlet processes. Technical Report 653, Department of Statistics, University of California, Berkeley, 2004.

[Thorisson *et al.*, 2005]  G.A. Thorisson, A.V. Smith, L. Krishnan, and L.D. Stein. The international hapmap project web site. *Genome Research*, 15:1591–1593, 2005.

[Xing *et al.*, 2004]  E.P. Xing, R. Sharan, and M.I Jordan. Bayesian haplotype inference via the Dirichlet process. In *Proceedings of the 21st International Conference on Machine Learning*, 2004.

[Xing *et al.*, 2006]  E.P. Xing, K.-A. Sohn, M.I Jordan, and Y. W. Teh. Bayesian multi-population haplotype inference via a hierarchical dirichlet process mixture. In *Proceedings of the 23st International Conference on Machine Learning*, 2006.

[Xing and Sohn, 2006]  E.P. Xing and K.-A. Sohn. Hidden Markov Dirichlet Process: Modeling Genetic Recombination in Open Ancestral Space. In *Bayesian Analysis*, to appear, 2007.

[Zhang *et al.*, 2002]  K. Zhang, M. Deng, T. Chen, M. Waterman, and F. Sun. A dynamic programming algorithm for haplotype block partitioning. *Proc. Natl. Acad. Sci. USA*, 99(11):7335–39, 2002.
